# VLSI Implementations of Learning and Memory Systems: A Review

**Mark A. Holler**
Intel Corporation
2250 Mission College Blvd.
Santa Clara, Ca. 95052-8125

## ABSTRACT

A large number of VLSI implementations of neural network models have been reported. The diversity of these implementations is noteworthy. This paper attempts to put a group of representative VLSI implementations in perspective by comparing and contrasting them. Design trade-offs are discussed and some suggestions for the direction of future implementation efforts are made.

## IMPLEMENTATION

Changing the way information is represented can be beneficial. For example a change of representation can make information more compact for storage and transmission. Implementation of neural computational models is just the process of changing the representation of a neural model from mathmatical symbolism to a physical embodiement for the purpose of shortening the time it takes to process information according to the neural model.

## FLEXIBILITY VS. PERFORMANCE

Today most neural models are already implemented in silicon VLSI, in the form of programs running on general purpose digital von Neumann computers. These machines are available at low cost and are highly flexible. Their flexibility results from the ease with which their programs can be changed. Maximizing flexibility, however, usually results in reduced performance. A program will often have to specify several simple op-

erations to carry out one higher level operation. An example is performing a sequence of shifts and adds to accomplish a multiplication. Higher level functions can be directly implemented but more hardware is required and that hardware can't be used to execute other high level functions. Flexibility is lost. This trade-off between flexibility and performance is a fundamental issue in computational device design and will be observed in the devices reviewed here.

## GROUND RULES

The neural network devices which will be discussed each consist of a set of what could loosely be called "artificial neurons". The artificial neurons typically calculate the inner product of an input vector and a stored weight vector, a sum of products of inputs times weights. An artificial synapse stores one weight and calculates one product or connection each time a new input is provided. The basic unit of computation is a "connection" and the basic measure of performance is the number of connections the neural network can perform per second, (CPS). The CPS number is directly related to how fast the chip will be able to perform mappings from input to output or recognize input patterns. The artificial neurons also include a non-linear thresholding function.

The comparison done here is restricted to devices which fit within this definition and as a result a number of important neural devices such as those which perform early vision processing or dynamical processing for optimization are not considered. In the interest of brevity only representative state of the art devices are presented.

## COMPARISON CRITERIA

The criteria for comparison are based on what would be important to a user: *Performance, Capability, Cost, Flexibility/ Ease of Application.*

In addition to the CPS measure of performance there is also a measure of how fast a chip can learn. How many connection or weight updates the chip can calculate and store per second, (CUPS) is an important performance measure for the chips which have learning capability. Three of the nine chips examined have learning on chip.

Important capabilities to consider are how big a network the chip can simulate, what precision of calculation the chip provides and how independent the chip is during learning. Table 1 provides neuron and synapse counts which indicate the maximum size network each chip can implement. The synaptic function and precision are noted in another column and comments about learning capability are also provided.

An interesting figure of merit is the ratio of CPS to the number of weights. This CPS per weight ratio will be referred to as the CPSPW. This figure of merit varies by over a factor of 1000 for the 9 chips considered and all have ratios much higher than typical von Neumann machines or the human brain. See the last column of Table 1. The significance of this disparity will be discussed later.

# TABLE 1. VLSI Neural Network Implementations

| | CPS | Connect Type | CUPS | Learning Algorithm | Neurons | Synapses | Technology | Weights | Config. | Avail. | Price | Synapse Area u² | CPSPW |
|---|---|---|---|---|---|---|---|---|---|---|---|---|---|
| Micro Devices [1] MD1220 Neural Bit Slice | 0.01B | 1b x 16b Product | NA | Off chip | 8 | 2048 | ? | Ext. | Board Level | Avail. | $45 | 5100 | 4883 |
| H.Graf, D.Henderson,[2] AT&T Bell Labs | 80B | 1b x 1-4b Product | NA | Off chip | 256 | 8K-32K | .9u CMOS | No | Chip Level | Board in 92 | ? | 1760 | 2.5-10M |
| Alspector, J., Allen, R.[3] Jayakumar, A., Bellcore | 0.1B | 5b x 5b Product | .1B | Boltzmann | 32 | 992 | 1.2u CMOS | No | Board Level | No | Reseach | 58344 | 100806 |
| Arima, Y., et al [4] Mitsubishi Electric | 5.6B | 1b x 6b product | 1.4B | Boltzmann | 336 | 28000 | 1u CMOS | No | Chip Level | No | Research | 4900 | 200000 |
| Hammerstrom, D., et al,[5] Adaptive Solutions | 1.6B | 1-16b x 1-16b multiple | .24B | Many; Back-Prop etc. | 64 | 128K-2M | .8u CMOS Multi-Field die | No | Chip Level | No | NA | 1400 | 800-3.1K |
| Agranat, R., et al [6] Ca. Inst. Tech. | 0.5B | 5b x 5b product | NA | Off chip | 256 | 65536 | 2u CMOS CCD | No | Board Level | No | Research | 560 | 7629 |
| Yasunaga, M., et al [7] Hitach; Wafer Scale | 2.3B | 6b x 6b product | NA | Off wafer | 1152 | 73700 | .8u CMOS Gate Array 8- 5" wafers | No | Wafer Level | No | >$10K | 410000 | 31208 |
| Tomlinson, M., et al [8] Neural Semiconductor | 0.1B | 4b x 4b product | NA | Off chip | 32 | 1024 | 1.2u CMOS | No | Board Level | 4/91 | $900 | 23000 | 97656 |
| Holler, M.,et al [9] Intel Corp. 80170, ETANN | 2B | 6b x 6b product | NA | Off chip | 64 | 10240 | 1u CMOS EEPROM | Non-Vol. | Chip Level | Avail. w/Tools | $940 | 2009 | 195313 |
| | | | | | | | | | | | | Brain | 100 |
| | | | | | | | | | | | | PC | 1 |

In addition to pricing information, what little exists, Table 1 includes the effective synapse area and process technology to give some indication of the relative cost of the various designs.

Finally to include something which suggests how flexible the chips are, comments are included in Table 1 to indicate whether or not the synaptic function, the learning algorithm and the network architecture, of each chip can be changed. Also to be considered is how hard it is to set or continuously refresh weights whether analog or digital. Analog vs. Digital I/O is a consideration as is availability and development tools. Demonstration in real applications would be another indicator of success but, none of these chips has yet reached this milestone.

## COMPARISON

The first device[1], from Micro Devices, is a digital neural network which leaves the weight memory off chip. Its eight 16 bit by one bit serial synapse multipliers are multiplexed which keeps the effective synapse cell size down. Using a single synaptic multiplier per neuron makes the total compute time for a neuron's sum of products dependent on how many inputs are supplied to a neuron. One positive aspect of this architecture is that any arbitrary number of inputs per neuron can be processed as long as the neuron accumulator is wide enough not to overflow when a worst case large sum of products is accumulated.

The Micro Devices chip shares this multiplexed synapse approach with the Adaptive Solutions X1 [5]and the CCD based design [6] by Agranat et al at Cal-Tech although these two chips include weight memory on chip to attain much better data transfer performance from the weight store to the synaptic processors. The multiplexed synapse approach is a good one for reducing the effective synapse size as can be seen by comparing the synapse area for these three chips[1,5,6] to those of the other chips. [5,6] have the two smallest cell sizes.

Micro Devices was first to introduce a commercial neural network chip and developent tools. They also have the lowest cost chip available. Its all digital interface makes it easy to design in. It's only significant limitations are its low neuron count and the fact that it can only accept binary inputs and output binary activations.

Hitachi's wafer scale neural network[7] designed with gate array technology uses pulse stream data representations as does the Neural Semiconductor implementation[8]. Pulse stream representations make the implementation of a digital multiplier trivial. It becomes just an AND gate. One drawback of this approach is that the user must convert his input data to uncorrelated pulse streams. The Hitachi design is also interesting because it is clearly designed to take advantage of the fault tolerant aspect of neural networks. The system they have built consists of eight wafers which are very likely to have at least several bad die. The automated gate array design used in the Hitachi resulted in the largest synapse area at 410,000 $u^2$.

Neural Semiconductor's design puts the neuron units on a separate chip from its synaptic units. This allows variable width input vectors with a large upper bound.

The CCD based design from Cal-Tech[6] is most noteworthy for its small cell size, 560 $u^2$, and high synapse count which results from the use of multiplexed synaptic processors and analog storage in a CCD. The drawback of this type of weight storage is that it must be refreshed every few milliseconds at higher temperatures.

Intel's 80170 [9] uses analog non-volatile weight storage and uses a basic characteristic of neural networks to advantage. It uses the adaptation that is going on during learning to adapt to variations in the analog circuit computing elements on the chip. This is noteworthy because it is another example of putting one of the properties of neural networks to use to enable a design approach different from conventional digital VLSI design.

The AT&T chip reported by Graf & Henderson [2] has achieved the highest CPS rate, 80B, of any of the chips. It was designed with handprinted character recognition in mind and as a result accepts only binary inputs (black and white). It uses a hybrid circuit design approach, digital for inputs and weight storage but analog summation in the form of currents. This chip is flexible in that its weight precision can be traded off for higher synapse count.

The last three chips[3,4,5] all have learning on chip. Two of them use Boltzmann learning which has been shown by Hinton [11] to be a form of gradient decent learning like Back-Propagation. These are the Bellcore chip[3] reported by J. Alspector, R. Allen and A. Jayakumar and the chip reported by Y. Arima et al at Mitsubishi[4]. The Misubishi chip has the most impressive number for learning performance and the second best mapping performance at 5.6B CPS. Its one drawback is that the analog weights it learns are volatile and must be refreshed. Bellcore's Boltzmann machine uses digital weight storage which does not require refresh. However, as you will notice the Bellcore synapse cell size is 10X larger due to the use of digital storage and a slightly lower density 1.2u technology.

The Adaptive Solutions chip with programmable learning and programmable synaptic function represents the flexibility end of the performance/flexibility trade-off. It is a single instruction multiple data path (SIMD) von Neumann machine. Its 64 synaptic processors are multiplexed up to 4096 times for eight bit weights making the effective synapse cell size very small, 1400 $u^2$ in spite of using digital SRAM for weight storage and fully digital synapse processors. This chip has the second smallest cell size primarily due to its multiplexing of the synaptic processing elements and because it multiplexes them more times than any of the other designs.

## CONNECTIONS PER SECOND PER WEIGHT     (CPSPW)

The ratio of connections/second per weight can be estimated for biological systems to be on the order of 100 assuming one weight is stored in each synapse. If neurons are

firing 100 times per second then each of the synapses must be processing pulses about 100 times per second hence the CPSPW of 100. This number is clearly related to neuron firing rate.    Less obvious is how CPSPW might be connected with the precison of the biological computing elements and the time frame in which the whole network seeks to produce final results.

Following von Neumann's arguments[12] arithmetical error grows in proportion to the number of steps of processing. This is partly due to round off errors and partly due to amplification of errors that occur early in the calculations. Biological neurons have limited precision due to their analog nature. If their calculations are accurate to within 1% and they are involved in a calculation that involves propagation of results through 100 neurons in sequence then the accumulated error could be as high as 100% meaning that the answer could be competely wrong. Any further calculations using this result would be useless. In other words, a 100 step calculation is the longest calculation you might expect a biological system to attempt to do because of its limited precision. Since the time frame that biological systems are typically concerned with is around 1 second one might expect to see these biological systems executing about 100 operations for each processing element in this interval. This appears to be the case. Executing any more operations than this would produce meaningless results due to the accumulation of numerical error.

A rule of thumb which summarizes the suggested relationship between CPSPW, precision and the time frame of interest would be: *The number of connections executed per weight in the interval of interest should be equal to the dynamic range of the weights.* The dynamic range of a weight is just the inverse of its precision or the maximum possible weight value minus the minimum weight value divided by the smallest increment in a weight which has significance.

Motor control, vision, handwriting and speech recognition tasks all fall within the "human time frame". The rule of thumb suggests that if neural network implementations with limited precision weights are used to solve these problems then these systems are likely to work best with the same CPSPW as biological neural systems, around 100. Since all of the neural network implementations reviewed here have CPSPW's well above 100 we might conclude that they are not optimal for these human time frame tasks. They don't have enough weights relative to their processing power. Standard von Neumann computers have CPSPW's which are much lower than those of biological systems. The number of operations per second per word of memory in a typical von Neumann machine is around 1. Von Neumann machines today don't have enough processing power relative to their memory size to be optimal for executing neural network solutions to problems in the human time frame.

For systems where results are sought in a time frame shorter than the human time frame a higher CPSPW should be used according to the rule of thumb.    All of the designs reviewed here have a CPSPW much higher than 100. See the right most column of

Table-1. The AT&T chip [2] has a CPSPW ratio in the millions and many of the chips[3,4,7,8,9] have a ratio around 100,000.    Chips with low CPSPW's are the same chips that multiplex the synaptic processors [1,5,6]. They can store more weights because they don't replicate the synaptic processor for every weight. Their processing rates are lowered which also lowers their CPSPW because they have fewer synaptic processors working simultaneously. This is not particularly desirable but results in a better balance between processing power and memory for tasks which don't need to be done any faster than in the human time frame.

**FUTURE DIRECTION**

Von Neumann machines with their high degree of flexibility will continue to be critical in the near term as neural models continue their rapid evolution. Multiprocessor ( >10) von Neumann machines optimized for neural type calculations are sorely needed. One such device [5] is already on the horizon. Hennessy and Patterson's quantitative approach [10] to computer design would be appropriate.

Neural network implementations with more weights are needed for making further progress in solving the difficult "human time domain" problems of speech and vision. A 1B CPS machine with 10M weights is needed. . Devices which multiplex the synaptic processing elements appear to be the best candidates for accomplishing this goal. The challenge here is to keep the bandwidth high even after the weight cache is moved off chip.

Using DRAM or "floating gate" memory cells which normally store digital information to store analog information instead in the same space is an approach which can be used in conjunction with multiplexed analog synapse processors to achieve a 6-8X improvement in the number of weights per synaptic processor with little penalty in die area. This general direction is largely unexplored except for the CCD implementation done by Agranat et al. [6].

VLSI implementations with fully parallel processing synaptic arrays represent a new computational capability; higher performance than can be achieved by any other means with given power and space. The availability of this new computing capability will open up new applications but, will likely take time. The majority of chips reviewed in this paper fall into this category [2,3,4,7,8,9].

## SUMMARY

The VLSI implementations to date are mostly high performance devices with limited memory. An image of slugs crawling at the speed of sound comes to mind. There will be applications for these "supersonic slugs", but, they are unlikely to make VLSI neural networks a big business any time soon. Implementations with more flexibility or more storage relative to processing power seem to be needed.

References

[1] Yestrebsky, J., Basehore, P. , Reed, J., "Neural Bit SliceComputing Element", Product Ap. Note TP102600, Micro Devices, Orlando, Fla.

[2] Graf, H., Henderson, D., "A Reconfigurable CMOS Neural Network", *1990 Int'l Solid State Circuits Conference*, San Francisco, Ca.

[3] Alspector, J., Allen, R., Jayakumar, A., "Relaxation Networks for Large Supervised Learning Problems", *Advances in Neural Information Processing Systems 3,* 1991 San Mateo, Ca.: Morgan Kaufmann

[4] Arima, Y., et al, "336 Neuron 28K Synapse Self-Learning Neural Network Chip with Branch Neuron-Unit Architecture", *1991 IEEE Int'l Solid State Circuits Conference*

[5]Griffin, M., Hammerstrom, D., et al, "An 11 Million Transistor Digital Neural Network Execution Engine", *1991 IEEE Int'l Solid State Circuits Conference*

[6] Agranat, R., Neugebauer, C., Yariv, A., "A CCD Based Neural Network Integrated Circuit with 64K Analog Programmable Synapses", *Int'l Joint Conference on Neural Networks,* June 1990.

[7] Gold, M., "Hitachi Unveils Prototype Neural Computer", *EE-Times,* Dec. 3, 1990.

[8] SU3232, NU32 Data Sheet, Neural Semiconductor Inc. , Carlsbad, Ca.

[9] 80170NW Electrically Trainable Analog Neural Network, Data Sheet, Intel Corp. Santa Clara, Ca. May 1990.

[10] Hennessy, J.L., Patterson, D. A. , *Computer Architecture, A Quantitative Approach",* p17, Morgan Kaufmann, San Mateo, Ca., 1990

[11] Hinton, G., "Deterministic Boltzmann Learning performs Steepest Decent in Weight-Space".

[12] Von Neumann, John, *The Computer and the Brain*, p26, 78 Yale University Press, New Haven, 1958

[13] Tam, S. et al, "Learning on an Analog VLSI Neural Network Chip", *1990 IEEE Int'l Conf. on Systems, Man and Cybernetics.*